# Generative Affine Localisation and Tracking

**John Winn**    **Andrew Blake**
Microsoft Research Cambridge
Roger Needham Building
7 J. J. Thomson Avenue
Cambridge CB3 0FB, U.K
http://research.microsoft.com/mlp

## Abstract

We present an extension to the Jojic and Frey (2001) layered sprite model which allows for layers to undergo affine transformations. This extension allows for affine object pose to be inferred whilst simultaneously learning the object shape and appearance. Learning is carried out by applying an augmented variational inference algorithm which includes a global search over a discretised transform space followed by a local optimisation. To aid correct convergence, we use bottom-up cues to restrict the space of possible affine transformations. We present results on a number of video sequences and show how the model can be extended to track an object whose appearance changes throughout the sequence.

## 1  Introduction

Generative models provide a powerful and intuitive way to analyse images or video sequences. Because such models directly represent the process of image generation, it is straightforward to incorporate prior knowledge about the imaging process and to interpret results. Since the entire data set is modelled, generative models can give improved accuracy and reliability over feature-based approaches and they also allow for selection between models using Bayesian model comparison. Finally, it is possible to sample from generative models, for example, for the purposes of image or video editing.

One popular type of generative model represents images as a composition of layers [1] where each layer corresponds to the appearance and shape of an individual object or the background. If the generative model is expressed probabilistically, Bayesian learning and inference techniques can then be applied to reverse the imaging process and infer the shape and appearance of individual objects in an unsupervised fashion [2].

The difficulty with generative models is how to apply Bayesian inference efficiently. In a layered model, inference involves localising the pose of the layers in each image, which is hard because of the large space of possible object transformations that needs to be explored. Previously, this has been dealt with by imposing restrictions on the space of object transformations, such as allowing only similarity transformations [3]. Alternatively, if the images are known to belong to a video sequence, tracking constraints can be used to focus the search on a small area of transformation space consistent with a dynamic model of object motion [4]. However, even in a video sequence, this technique relies on the object

remaining in frame and moving relatively slowly.

In this paper, we extend the work of [3] and present an approach to object localisation which allows objects to undergo planar affine transformations and works well both in the frames of a video sequence and in unordered sets of images. A two-layer generative model is defined and inference performed using a factorised variational approximation, including a global search over a discretised transform space followed by a local optimisation using conjugate gradients. Additionally, we exploit bottom-up cues to constrain the space of transforms being explored. Finally, we extend our generative model to allow the object appearance in one image to depend on its appearance in the previous one. Tracking appearance in this way gives improved performance for objects whose appearance changes slowly over time (e.g. objects undergoing non-planar rotation).

If the images are not frames of a video, or the object is out-of-frame or occluded in the previous image, then the system automatically reverts to using a learned foreground appearance model.

## 2    The generative image model

This section describes the generative image model, which is illustrated in the Bayesian network of Figure 1. This model consists of two layers, a foreground layer containing a single object and a background layer.

We denote our image set as $\{\mathbf{x}_1, \ldots, \mathbf{x}_N\}$, where $\mathbf{x}_i$ is a vector of the pixel intensities in the $i$th image. The background layer is assumed to be stationary and so its appearance vector $\mathbf{b}$ is set to be the same size as the image. A mask $\mathbf{m}_i$ has binary elements that indicate which

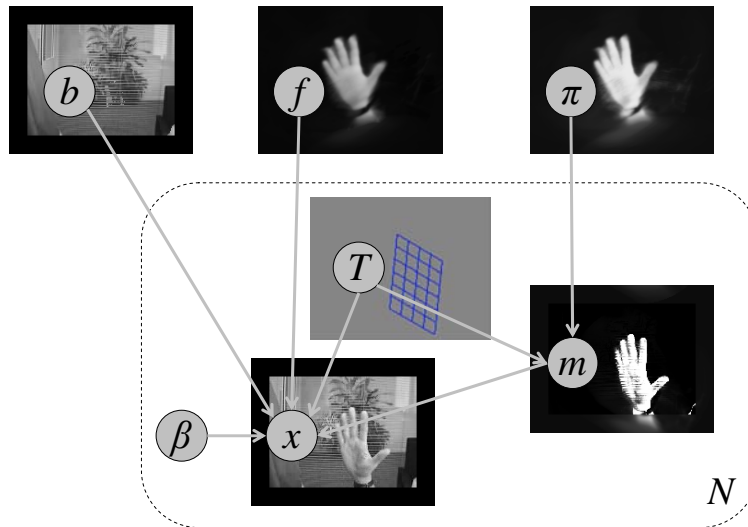

**Figure 1: The Bayesian network for the generative image model.** The rounded rectangle is a *plate*, indicating that there are $N$ copies of each contained node (one for each image). Common to all images are the background $\mathbf{b}$, foreground object appearance $\mathbf{f}$ and mask prior $\boldsymbol{\pi}$. An affine transform $\mathbf{T}$ gives the position and pose of the object in each image. The binary mask $\mathbf{m}$ defines the area of support of the foreground object and has a prior given by a transformed $\boldsymbol{\pi}$. The observed image $\mathbf{x}$ is generated by adding noise $\boldsymbol{\beta}$ separately to the transformed foreground appearance and the background and composing them together using the mask. For illustration, the images underneath each node of the graph represent the inferred value of that node given a data set of hand images. A priori, the appearance and mask of the object are not known.

pixels of the $i$th image are foreground. The mask is set to be slightly larger than the image to allow the foreground object to overlap the edge of the image.

The foreground layer is represented by an appearance image vector $\mathbf{f}$ and a prior over its mask $\boldsymbol{\pi}$, both of which are to be inferred from the image set. The elements of $\boldsymbol{\pi}$ are real numbers in the range $[0, 1]$ which indicate the probability that the corresponding mask pixels are on, as suggested in [5]. The object appearance and mask prior are defined in a canonical, normalised pose; the actual position and pose of the object in the $i$th image is given by an affine transformation $\mathbf{T}_i$. With our images in vector form, we can consider a transformation $\mathbf{T}$ to be a sparse matrix where the $j$th row defines the linear interpolation of pixels that gives the $j$th pixel in the transformed image. For example, a translation of an integer number of pixels is represented as a matrix whose entries $T_{jk}$ are 1 if the translation of location $k$ in the source image is location $j$ in the destination image, and 0 otherwise. Hence, the transformed foreground appearance is given by $\mathbf{Tf}$ and the transformed mask prior by $\mathbf{T}\boldsymbol{\pi}$. Given the transformed mask prior, the conditional distribution for the $k$th mask pixel is

$$P(m_k = 1 \,|\, \boldsymbol{\pi}, \mathbf{T}) = (\mathbf{T}\boldsymbol{\pi})_k. \tag{1}$$

The observed image $\mathbf{x}$ is generated by a composition of the transformed foreground appearance and the background plus some noise. The conditional distribution for the $k$th image pixel is given by

$$P(x_k \,|\, \mathbf{b}, \mathbf{f}, \mathbf{m}, \mathbf{T}, \boldsymbol{\beta}) = \mathcal{N}(x_k \,|\, (\mathbf{Tf})_k, \beta_f^{-1})^{m_k} \mathcal{N}(x_k \,|\, b_k, \beta_b^{-1})^{1-m_k} \tag{2}$$

where $\boldsymbol{\beta} = (\beta_f, \beta_b)$ are the noise precisions for the foreground layer and the background layer respectively. The elements of both $\mathbf{b}$ and $\mathbf{f}$ are given broad Gaussian priors and the prior on each $\beta$ is a broad Gamma distribution. The prior on each element of $\boldsymbol{\pi}$ is a Beta distribution.

## 3   Factorised variational inference

Given the above model and a set of images $\{\mathbf{x}_1, \ldots, \mathbf{x}_N\}$, the inference task is to learn a posterior distribution over all other variables including the background, the foreground appearance and mask prior, the transformation and mask for each image and the noise precisions. Direct application of Bayes's theorem is intractable because this would require integrating over all unobserved variables. Instead, we turn to the approximate inference technique of *variational inference* [6].

Variational inference involves defining a factorised variational distribution $Q$ and then optimising to minimise the Kullback-Leibler divergence between $Q$ and the true posterior distribution. The motivation behind this methodology is that we expect the posterior to be unimodal and tightly peaked and so it can be well approximated by a separable distribution. In this paper, we choose our variational distribution to be factorised with respect to each element of $\mathbf{b}$, $\mathbf{f}$, $\mathbf{m}$ and $\boldsymbol{\pi}$ and also with respect to $\beta_f$ and $\beta_b$. The factor of $Q$ corresponding to each of these variables has the same form as the prior over that variable. For example, the factor for the $k$th element of $\boldsymbol{\pi}$ is a Beta distribution $Q(\pi_k) = \beta(\pi_k \,|\, a', b')$. The choice of approximation to the posterior over the affine transform $Q(\mathbf{T})$ is a more complex one, and will be discussed below.

The optimisation of the $Q$ distribution is achieved by firstly initialising the parameters of all factors and then iteratively updating each factor in turn so as to minimise the KL divergence whilst keeping all other factors fixed. If we define $\mathbf{H}$ to be the set of all hidden variables, then the factor for the $i$th member $H_i$ is updated using

$$\log Q(H_i) = \langle \log P(\{\mathbf{x}_1, \ldots, \mathbf{x}_N\}, \mathbf{H}) \rangle_{\sim Q(H_i)} + \text{const.} \tag{3}$$

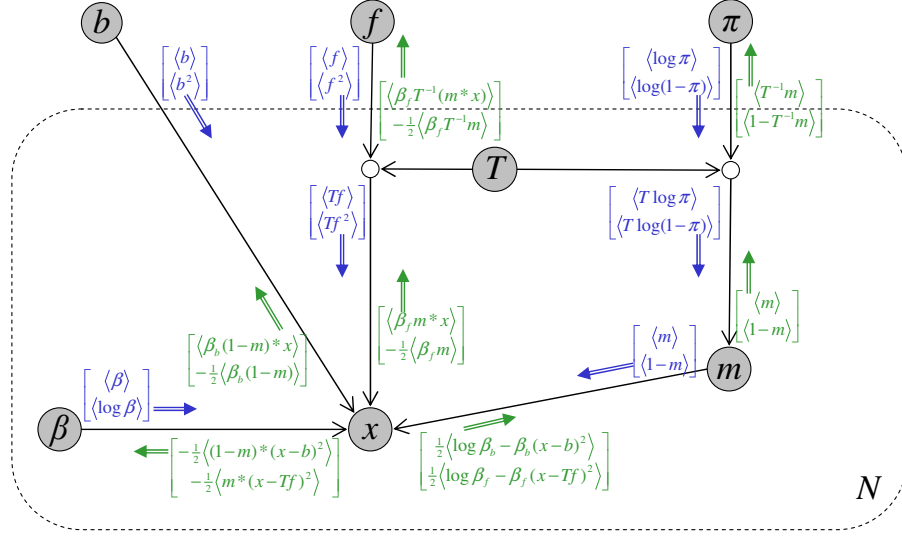

**Figure 2: The messages passed when VMP is applied to the generative model.** The messages to or from $\mathbf{T}$ are not shown (see text). Where a message is shown as leaving the $N$ plate, the destination node receives a set of $N$ messages, one from each copy of the nodes within the plate. Where a message is shown entering the $N$ plate, the message is sent to all copies of the destination node. All expectations are with respect to the variational distribution $Q$.

where $\langle.\rangle_{\sim Q(H_i)}$ means the expectation under the distribution given by the product of all factors of $Q$ except $Q(H_i)$.

When the model is a Bayesian network, this optimisation procedure can be carried out in a modular fashion by applying Variational Message Passing (VMP) [7, 8]. Using VMP makes it very much simpler and quicker to extend, modify, combine or compare probabilistic models; it gives the same results as applying factorised variational inference by hand and places no additional constraints on the form of the model. In VMP, messages consisting of vectors of real numbers are sent to each node from its parent and children in the graph. In our model, the messages to and from all nodes (except $\mathbf{T}$) are shown in Figure 2. By expressing each variational factor as an exponential family distribution, the 'natural parameter vector' [8] of that distribution can be optimised using (3) by adding messages received at the corresponding node. For example, if the prior over $\mathbf{b}$ is $\mathcal{N}(\mathbf{b}\,|\,\boldsymbol{\mu}, \boldsymbol{\gamma}^{-1})$, the parameter vector of the factor $Q(\mathbf{b}) = \mathcal{N}(\mathbf{b}\,|\,\boldsymbol{\mu}', \boldsymbol{\gamma}'^{-1})$ is updated from the messages received at $\mathbf{b}$ using

$$
\overbrace{\begin{bmatrix} \boldsymbol{\mu}'\boldsymbol{\gamma}' \\ -\frac{1}{2}\boldsymbol{\gamma}' \end{bmatrix}}^{\text{natural param. vector}} = \overbrace{\begin{bmatrix} \boldsymbol{\mu}\boldsymbol{\gamma} \\ -\frac{1}{2}\boldsymbol{\gamma} \end{bmatrix}}^{\text{prior}} + \sum_{i=1}^{N} \overbrace{\begin{bmatrix} \langle \beta_{bi}(1-\mathbf{m}_i) * \mathbf{x}_i \rangle \\ -\frac{1}{2}\langle \beta_{bi}(1-\mathbf{m}_i) \rangle \end{bmatrix}}^{\text{received messages}}. \tag{4}
$$

The form of the natural parameter vector varies for different exponential family distributions (Gaussian, Gamma, Beta, discrete . . . ) but the update equation remains the same. Following this update, the message being sent from $\mathbf{b}$ is recomputed to reflect the new parameters of $Q(\mathbf{b})$. For details of the derivation this update equation and how to determine VMP messages for a given model, see [8].

Where a set of similar messages are sent corresponding to the pixels of an image, it is convenient to think instead of a single message where each element is itself an image. It is efficient to structure the implementation in this way because message computation and parameter updates can then be carried out using block operations on entire images.

# 4 Learning the object transformation

Following [3], we decompose the layer transformation into a product of transformations and define a variational distribution that is separable over each. To allow for affine transformations, we choose to decompose $\mathbf{T}$ into three transformations applied sequentially,

$$\mathbf{T} = \mathbf{T}_{xy}\mathbf{T}_{rs}\mathbf{T}_a. \tag{5}$$

In this expression, $\mathbf{T}_{xy}$ is a two-dimensional translation belonging to a finite set of translations $\mathcal{T}_{xy}$. Similarly, $\mathbf{T}_{rs}$ is a rotation and uniform scaling and so the space of transforms is also two-dimensional and is discretised to form a finite set $\mathcal{T}_{rs}$. The third transformation $\mathbf{T}_a$ is a freeform (non-discretised) affine transform. The variational distribution over the combined transform $\mathbf{T}$ is given by

$$Q(\mathbf{T}) = Q(\mathbf{T}_{xy})Q(\mathbf{T}_{rs})Q(\mathbf{T}_a). \tag{6}$$

Because $\mathbf{T}_{xy}$ and $\mathbf{T}_{rs}$ are discretised, $Q(\mathbf{T}_{xy})$ and $Q(\mathbf{T}_{rs})$ are defined to be discrete distributions. We can apply (3) to determine the update equations for these distributions,

$$
\begin{aligned}
\log Q(\mathbf{T}_{xy}) &= \langle \mathbf{m} \rangle . (\mathbf{T}_{xy} \langle \mathbf{T}_{rs}\mathbf{T}_a \log \boldsymbol{\pi} \rangle) + \langle 1 - \mathbf{m} \rangle . (\mathbf{T}_{xy} \langle \mathbf{T}_{rs}\mathbf{T}_a \log(1 - \boldsymbol{\pi}) \rangle) \\
&\quad + \beta_f \langle \mathbf{m} \rangle . \left( \mathbf{x} * \mathbf{T}_{xy} \langle \mathbf{T}_{rs}\mathbf{T}_a \mathbf{f} \rangle - \tfrac{1}{2}\mathbf{T}_{xy} \langle \mathbf{T}_{rs}\mathbf{T}_a \mathbf{f}^2 \rangle \right) + z^{xy} \tag{7}
\end{aligned}
$$

$$
\begin{aligned}
\log Q(\mathbf{T}_{rs}) &= \left\langle \mathbf{T}_{xy}^{-1}\mathbf{m} \right\rangle . (\mathbf{T}_{rs} \langle \mathbf{T}_a \log \boldsymbol{\pi} \rangle) + \left\langle \mathbf{T}_{xy}^{-1}(1 - \mathbf{m}) \right\rangle . (\mathbf{T}_{rs} \langle \mathbf{T}_a \log(1 - \boldsymbol{\pi}) \rangle) \\
&\quad + \beta_f \left\langle \mathbf{T}_{xy}^{-1}(\mathbf{m} * \mathbf{x}) \right\rangle . (\mathbf{T}_{rs} \langle \mathbf{T}_a \mathbf{f} \rangle) - \tfrac{1}{2}\beta_f \left\langle \mathbf{T}_{xy}^{-1}\mathbf{m} \right\rangle \left( \mathbf{T}_{rs} \left\langle \mathbf{T}_a \mathbf{f}^2 \right\rangle \right) + z^{rs} \tag{8}
\end{aligned}
$$

where $z^{xy}$ and $z^{rs}$ are constants which can be found by normalisation.

As described in [3], the evaluation of (7) and (8) for all $\mathbf{T}_{xy} \in \mathcal{T}_{xy}$ and all $\mathbf{T}_{rs} \in \mathcal{T}_{rs}$ can be carried out efficiently using Fast Fourier Transforms in either Cartesian or log-polar co-ordinate systems. The use of FFTs allows us to make both $\mathcal{T}_{xy}$ and $\mathcal{T}_{rs}$ large: we set $\mathcal{T}_{xy}$ to contain all translations of a whole number of pixels and $\mathcal{T}_{rs}$ to contain 360 rotations (at $1°$ intervals) and 50 scalings (where each scaling represents a $\sim 1.5\%$ increase in length scale). FFTs can be used within the VMP framework as both (7) and (8) involve quantities that are contained in messages to $\mathbf{T}$ (see Figure 2).

Finally, we define the variational distribution over $\mathbf{T}_a$ to be a delta function,

$$Q(\mathbf{T}_a) = \delta(\mathbf{T}_a - \mathbf{T}_a^\star). \tag{9}$$

Unlike all the other variational factors, this cannot be optimised analytically. To minimise the KL divergence, we need to find the value of $\mathbf{T}_a^\star$ that maximises

$$
\begin{aligned}
\mathcal{F}_a &= \left\langle \mathbf{T}_{rs}^{-1}\mathbf{T}_{xy}^{-1}\mathbf{m} \right\rangle . (\mathbf{T}_a^\star \langle \log \boldsymbol{\pi} \rangle) + \left\langle \mathbf{T}_{rs}^{-1}\mathbf{T}_{xy}^{-1}(1 - \mathbf{m}) \right\rangle . (\mathbf{T}_a^\star \langle \log(1 - \boldsymbol{\pi}) \rangle) \\
&\quad + \beta_f \left\langle \mathbf{T}_{rs}^{-1}\mathbf{T}_{xy}^{-1}(\mathbf{m} * \mathbf{x}) \right\rangle . (\mathbf{T}_a^\star \langle \mathbf{f} \rangle) - \tfrac{1}{2}\beta_f \left\langle \mathbf{T}_{rs}^{-1}\mathbf{T}_{xy}^{-1}\mathbf{m} \right\rangle (\mathbf{T}_a^\star \langle \mathbf{f}^2 \rangle) . \tag{10}
\end{aligned}
$$

This local maximisation is achieved efficiently by using a trust-region Newton method. The assumption is that the search through $\mathcal{T}_{xy}$ and $\mathcal{T}_{rs}$ has located the correct posterior mode in transform space and that it is only necessary to use gradient methods to find the peak of that mode. This assumption appeared valid for the image sequences used in our experiments, even when the transformation of the foreground layer was not well approximated by a similarity transform alone.

Inference in this model is made harder due to an inherent non-identifiability problem. The pose of the learned appearance and mask prior is undefined and so applying a transform to $\mathbf{f}$ and $\boldsymbol{\pi}$ and the inverse of the transform to each $\mathbf{T}_i$ results in an unchanged joint distribution. When applying a variational technique, such non-identifiability leads to many more local minima in the KL divergence. We partially resolve this issue by adding a constraint to this model that the expected mask $\langle \boldsymbol{\pi} \rangle$ is centred, so that its centre of gravity is in the middle of the latent image. This constraint is applied by shifting the parameters of $Q(\boldsymbol{\pi})$ directly following each update (and also shifting $Q(\mathbf{f})$ and each $Q(\mathbf{T})$ appropriately).

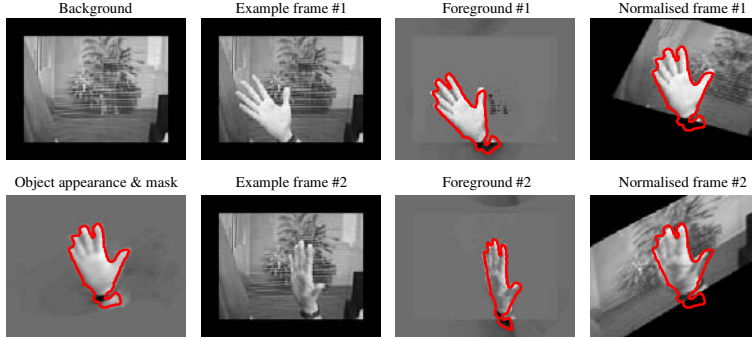

| Background | Example frame #1 | Foreground #1 | Normalised frame #1 |

| Object appearance & mask | Example frame #2 | Foreground #2 | Normalised frame #2 |

**Figure 3: Tracking a hand undergoing extreme affine transformation.** The first column shows the learned background and masked object appearance. The second and third columns contain two frames from the sequence along with the foreground segmentation for each. The final column shows each frame transformed by the inverse of the inferred object transform. In each image the red outline surrounds the area where the transformed mask prior $\boldsymbol{\pi}$ is greater than $0.5$.

## 4.1 Using bottom-up information to improve inference

Given that $\boldsymbol{\pi}$ is centred, we can significantly improve convergence by using bottom-up information about the translation of the object. For example, the inferred mask $\mathbf{m}_i$ for each frame is very informative about the location of the object in that frame. Using sufficient data, we could learn a conditional model $P(\mathbf{T}_{xy} \mid \langle \mathbf{m}_i \rangle)$ and bound $\mathbf{T}_{xy}$ by only considering translations with non-negligible posterior mass under this conditional model. Instead, we use a conservative, hand-constructed bound on $\mathbf{T}_{xy}$ based on the assumption that, during inference, the most probable mask under $Q(\mathbf{m}_i)$ consists of a (noisy) subset of the true mask pixels. Suppose the true mask contains $M$ non-zero pixels with second moment of area $\mathbf{I}_M$ and the current most probable mask contains $V$ non-zero pixels ($V \leq M$) with second moment of area $\mathbf{I}_V$. A bound on $\mathbf{c}$, the position of the centre of the inferred mask relative to the centre of true mask, is given by

$$\operatorname{diag}(\mathbf{c}\mathbf{c}^{\mathrm{T}}) \leq (M - V)\operatorname{diag}(\mathbf{I}_M/V - \mathbf{I}_V/M). \tag{11}$$

We can gain a conservative estimate of $M$ and $\mathbf{I}_M$ by using the maximum values of $V$ and $\mathbf{I}_V$ across all frames, multiplied by a constant $\alpha \approx 1.2$. The bound is deliberately constructed to be conservative; its purpose is to discard settings of $\mathbf{T}_{xy}$ that have negligible probability under the model and so avoid local minima in the variational optimisation. The bound is updated at each iteration and applied by setting $Q(\mathbf{T}_{xy}) = 0$ for values of $\mathbf{T}_{xy}$ outside the bound. $Q(\mathbf{T}_{xy})$ is then re-normalised.

The use of this bound on $\mathbf{T}_{xy}$ is intended as a very simple example of incorporating bottom-up information to improve inference within a generative model. In future work, we intend to investigate using more informative bottom-up cues, such as optical flow or tracked interest points, to propose probable transformations within this model. Incorporating such proposals or bounds into a variational inference framework both speeds convergence and helps avoid local minima.

## 5 Experimental results

We present results on two video sequences. The first is of a hand rotating both parallel to the image plane and around its own axis, whilst also translating in three dimensions. The sequence consists of 59 greyscale frames, each of size $160 \times 120$ pixels (excluding the border). Our Matlab implementation took about a minute per frame to analyse the se-

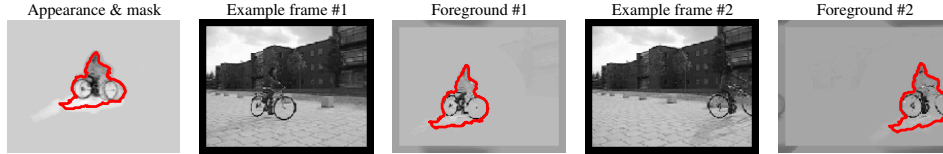

| Appearance & mask | Example frame #1 | Foreground #1 | Example frame #2 | Foreground #2 |

**Figure 4: Affine tracking of a semi-transparent object.**

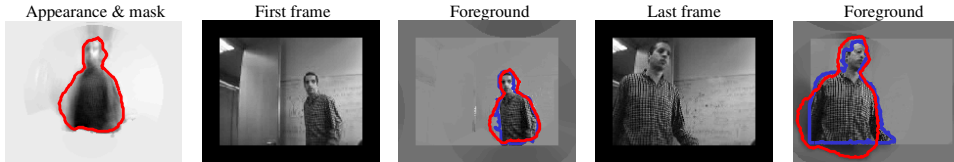

| Appearance & mask | First frame | Foreground | Last frame | Foreground |

**Figure 5: Tracking an object with changing appearance.** A person is tracked throughout a sequence despite their appearance changing dramatically from between first and last frames. The blue outline shows the inferred mask $\mathbf{m}$ which differs slightly from $\boldsymbol{\pi}$ due to the object changing shape.

quence, over half of which was spent on the conjugate gradient optimisation step. Figure 3 shows the expected values of the background and foreground layers under the optimised variational distribution, along with foreground segmentation results for two frames of the sequence. The right hand column gives another indication of the accuracy of the inferred transformations by applying the inverse transformation to the entire frame and showing that the hand then has a consistent normalised position and size. In a video of the hand showing the tracked outline,[1] the outline appears to move smoothly and follow the hand with a high degree of accuracy, despite the system not using any temporal constraints.

Results for a second sequence showing a cyclist are given in Figure 4. Although the cyclist and her shadow are tracked correctly, the learned appearance is slightly inaccurate as the system is unable to capture the perspective foreshortening of the bicycle. This could be corrected by allowing $\mathbf{T}_a$ to include projective transformations.

## 6 Tracking objects with changing appearance

The model described so far makes the assumption that the appearance of the object does not change significantly from frame to frame. If the set of images are actually frames from a video, we can model objects whose appearance changes slowly by allowing the model to use the object appearance in the previous frame as the basis for its appearance in the current frame. However, we may not know if the images are video frames and, even if we do, the object may be occluded or out-of-frame in the previous image. We can cope with this uncertainty by inferring automatically whether to use the previous frame or the learned appearance $\mathbf{f}$. Switching between two methods in this way is similar to [9].

The model is extended by introducing a binary variable $s_i$ for each frame and define a new appearance variable $\mathbf{g}_i = s_i \mathbf{f} + (1 - s_i)\mathbf{T}_{i-1}^{-1}\mathbf{x}_{i-1}$. Hence $\mathbf{g}_i$ either equals the foreground appearance $\mathbf{f}$ (if $s_i = 1$) or the transform-normalised previous frame (if $s_i = 0$). For the first frame, we fix $s_1 = 1$. We then replace $\mathbf{f}$ with $\mathbf{g}_i$ in (2) and then apply VMP within the resulting Bayesian network.

The extended model is able to track an object even when its appearance changes significantly throughout the image sequence (see Figure 5). The binary variable $s_i$ is found to have an expected value $\approx 0$ for all frames (except the first). Using the tracked appearance

allows the foreground segmentation of each frame to be accurate even though the object is poorly modelled by the inferred appearance image. If we introduce an abrupt change into the sequence, for example by reversing the second half of the sequence, $\langle s_i \rangle$ is found to be $\approx 1$ for the frame following the change. In other words, the system has detected not to use the previous frame at this point, but to revert to using the latent appearance image $\mathbf{f}$.

## 7   Discussion

We have proposed a method for localising an object undergoing affine transform whilst simultaneously learning its shape and appearance. This power of this method has been demonstrated by tracking moving objects in several real videos, including where the appearance of the object changes significantly from start to end. The system makes no assumptions about the speed of motion of the object, requires no special initialisation and is robust to the object being temporarily occluded or moving out of frame.

A natural extension to this work is to allow multiple layers, with each layer having its own latent shape and appearance and set of affine transformations. Unfortunately, as the number of latent variables increases, the inference problem becomes correspondingly harder and an exhaustive search becomes less practical. Instead, we are investigating performing inference in a simpler model where a subset of the variables have been approximately marginalised out. The results of using this simpler model can then be used to guide inference in the full model. A further interesting addition to the model would be to allow layers to be grouped into rigid or articulated three-dimensional objects.

### Acknowledgments

The authors would like to thank Nebojsa Jojic for suggesting the use of a binary switch variable for tracking and Tom Minka for helpful discussions.

## Footnotes

[1]Videos of results are available from `http://johnwinn.org/Research/affine`.

## References

[1] J. Y. A. Wang and E. H. Adelson. Representing moving images with layers. In *IEEE Transactions on Image Processing*, volume 3, pages 625–638, 1994.

[2] N. Jojic and B. Frey. Learning flexible sprites in video layers. In *Proc. of IEEE Conf. on Computer Vision and Pattern Recognition*, 2001.

[3] B. Frey and N. Jojic. Fast, large-scale transformation-invariant clustering. In *Advances in Neural Information Processing Systems 14*, 2001.

[4] M. K. Titsias and C. K. I. Williams. Fast unsupervised greedy learning of multiple objects and parts from video. 2004. To appear in Proc. Generative-Model Based Vision Workshop, Washington DC, USA.

[5] C.K.I. Williams and M. K. Titsias. Greedy learning of multiple objects in images using robust statistics and factorial learning. *Neural Computation*, 16(5):1039–1062, 2004.

[6] M. I. Jordan, Z. Ghahramani, T. S. Jaakkola, and L. K. Saul. An introduction to variational methods for graphical models. In M. I. Jordan, editor, *Learning in Graphical Models*, pages 105–162. Kluwer, 1998.

[7] C. M. Bishop, J. M. Winn, and D. Spiegelhalter. VIBES: A variational inference engine for Bayesian networks. In *Advances in Neural Information Processing Systems*, volume 15, 2002.

[8] J. M. Winn and C. M. Bishop. Variational Message Passing. 2004. To appear in Journal of Machine Learning Research. Available from `http://johnwinn.org`.

[9] A. Jepson, D. Fleet, and T. El-Maraghi. Robust online appearance models for visual tracking. In *Proc. IEEE Conf. Computer Vision and Pattern Recognition*, volume I, pages 415–422, 2001.
